# Regularized estimation of image statistics by Score Matching

**Diederik P. Kingma**
Department of Information and Computing Sciences
Universiteit Utrecht
`d.p.kingma@students.uu.nl`

**Yann LeCun**
Courant Institute of Mathematical Sciences
New York University
`yann@cs.nyu.edu`

## Abstract

Score Matching is a recently-proposed criterion for training high-dimensional density models for which maximum likelihood training is intractable. It has been applied to learning natural image statistics but has so-far been limited to simple models due to the difficulty of differentiating the loss with respect to the model parameters. We show how this differentiation can be automated with an extended version of the double-backpropagation algorithm. In addition, we introduce a regularization term for the Score Matching loss that enables its use for a broader range of problem by suppressing instabilities that occur with finite training sample sizes and quantized input values. Results are reported for image denoising and super-resolution.

## 1 Introduction

Consider the subject of *density estimation* for high-dimensional continuous random variables, like images. Approaches for normalized density estimation, like mixture models, often suffer from the curse of dimensionality. An alternative approach is *Product-of-Experts* (PoE) [7], where we model the density as a *product*, rather than a sum, of component (*expert*) densities. The multiplicative nature of PoE models make them able to form complex densities: in contrast to mixture models, each expert has the ability to have a strongly negative influence on the density at any point by assigning it a very low component density. However, Maximum Likelihood Estimation (MLE) of the model requires differentiation of a normalizing term, which is infeasible even for low data dimensionality.

A recently introduced estimation method is Score Matching [10], which involves minimizing the square distance between the model log-density slope (*score*) and data log-density slope, which is independent of the normalizing term. Unfortunately, applications of SM estimation have thus far been limited. Besides ICA models, SM has been applied to Markov Random Fields [14] and a multi-layer model [13], but reported results on real-world data have been of qualitative, rather than quantitative nature. Differentiating the SM loss with respect to the parameters can be very challenging, which somewhat complicates the use of SM in many situations. Furthermore, the proof of the SM estimator [10] requires certain conditions that are often violated, like a smooth underlying density or an infinite number of samples.

Other estimation methods are Constrastive Divergence [8] (CD), Basis Rotation [23] and Noise-Contrastive Estimation [6] (NCE). CD is an MCMC method that has been succesfully applied to Restricted Boltzmann Machines (RBM's) [8], overcomplete Independent Component Analysis

(ICA) [9], and convolution variants of ICA and RBM's [21, 19]. Basis Rotation [23] works by restricting weight updates such that they are probability mass-neutral. SM and NCE are consistent estimators [10, 6], while CD estimation has been shown to be generally asymptotically biased [4]. No consistency results are known for Basis Rotation, to our knowledge. NCE is a promising method, but unfortunately too new to be included in experiments. CD and Basis Rotation estimation will be used as a basis for comparison.

In section 2 a regularizer is proposed that makes Score Matching applicable to a much broader class of problems. In section 3 we show how computation and differentiation of the SM loss can be performed in automated fashion. In section 4 we report encouraging quantitative experimental results.

## 2 Regularized Score Matching

Consider an energy-based [17] model $E(\mathbf{x}; \mathbf{w})$, where "energy" is the unnormalized negative log-density such that the pdf is: $p(\mathbf{x}; \mathbf{w}) = e^{-E(\mathbf{x}; \mathbf{w})}/Z(\mathbf{w})$, where $Z(\mathbf{w})$ is the normalizing constant. In other words, low energies correspond to high probability density, and high energies correspond to low probability density.

Score Matching works by fitting the slope (*score*) of the model density to the slope of the true, underlying density at the data points, which is obviously independent of the vertical offset of the log-density (the normalizing constant). Hyvärinen [10] shows that under some conditions, this objective is equivalent to minimizing the following expression, which involves only first and second partial derivatives of the model density:

$$J(\mathbf{w}) = \int_{\mathbf{x} \in \mathbb{R}^N} p_{\mathbf{x}}(\mathbf{x}) \sum_{i=1}^{N} \left( \frac{1}{2} \left( \frac{\partial E(\mathbf{x}; \mathbf{w})}{\partial x_i} \right)^2 - \frac{\partial^2 E(\mathbf{x}; \mathbf{w})}{(\partial x_i)^2} \right) d\mathbf{x} + const \qquad (1)$$

with $N$-dimensional data vector $\mathbf{x}$, weight vector $\mathbf{w}$ and true, underlying pdf $p_{\mathbf{x}}(\mathbf{x})$. Among the conditions [1] is (1) that $p_{\mathbf{x}}(\mathbf{x})$ is differentiable, and (2) that the log-density is finite everywhere. In practice, the true pdf is unknown, and we have a finite sample of $T$ discrete data points. The sample version of the SM loss function is:

$$J^S(\mathbf{w}) = \frac{1}{T} \sum_{t=1}^{T} \sum_{i=1}^{N} \left( \frac{1}{2} \left( \frac{\partial E(\mathbf{x}_{(t)}; \mathbf{w})}{\partial x_i} \right)^2 - \frac{\partial^2 E(\mathbf{x}_{(t)}; \mathbf{w})}{(\partial x_i)^2} \right) \qquad (2)$$

which is asymptotically equivalent to the equation (1) as $T$ approaches infinity, due to the law of large numbers. This loss function was used in previous publications on SM [10, 12, 13, 15].

### 2.1 Issues

Should these conditions be violated, then (theoretically) the pdf cannot be estimated using equation (1). Only some specific special-case solutions exist, e.g. for non-negative data [11]. Unfortunately, situations where the mentioned conditions are violated are not rare. The distribution for quantized data (like images) is discontinuous, hence not differentiable, since the data points are concentrated at a finite number of discrete positions. Moreover, the fact that equation (2) is only equivalent to equation (1) as $T$ approaches infinity may cause problems: the distribution of any finite training set of discrete data points is discrete, hence not differentiable. For proper estimation with SM, data can be smoothened by whitening; however, common whitening methods (such as PCA or SVD) are computational infeasible for large data dimensionality, and generally destroy the local structure of spatial and temporal data such as image and audio. Some previous publications on Score Matching apply zero-phase whitening (ZCA) [13] which computes a weighed sum over an input patch which removes some of the original quantization, and can potentially be applied convolutionally. However,

the amount of information removed from the input by such whitening is not parameterized and potentially large.

## 2.2 Proposed solution

Our proposed solution is the addition of a regularization term to the loss, approximately equivalent to replacing each data point $\mathbf{x}$ with a Gaussian cloud of virtual datapoints $(\mathbf{x}+\epsilon)$ with i.i.d. Gaussian noise $\epsilon \sim \mathcal{N}(\mathbf{0}, \sigma^2 I)$. By this replacement, the sample pdf becomes smooth and the conditions for proper SM estimation become satisfied. The expected value of the sample loss is:

$$\mathbb{E}\left[J^S(\mathbf{x}+\epsilon; \mathbf{w})\right] = \frac{1}{2}\sum_{i=1}^{N}\left(\mathbb{E}\left[\left(\frac{\partial E(\mathbf{x}+\epsilon; \mathbf{w})}{\partial(x_i+\epsilon_i)}\right)^2\right]\right) - \sum_{i=1}^{N}\left(\mathbb{E}\left[\frac{\partial^2 E(\mathbf{x}+\epsilon; \mathbf{w})}{(\partial(x_i+\epsilon_i))^2}\right]\right) \quad (3)$$

We approximate the first and second term with a simple first-order Taylor expansion. Recall that since the noise is i.i.d. Gaussian, $\mathbb{E}\left[\epsilon_i\right] = 0$, $\mathbb{E}\left[\epsilon_i\epsilon_j\right] = \mathbb{E}\left[\epsilon_i\right]\mathbb{E}\left[\epsilon_j\right] = 0$ if $i \neq j$, and $\mathbb{E}\left[\epsilon_i^2\right] = \sigma^2$. The expected value of the first term is:

$$
\begin{aligned}
\frac{1}{2}\sum_{i=1}^{N}\mathbb{E}\left[\left(\frac{\partial E(\mathbf{x}+\epsilon; \mathbf{w})}{\partial(x_i+\epsilon_i)}\right)^2\right] &= \frac{1}{2}\sum_{i=1}^{N}\mathbb{E}\left[\left(\frac{\partial E(\mathbf{x}; \mathbf{w})}{\partial x_i} + \sum_{j=1}^{N}\left(\frac{\partial^2 E(\mathbf{x}; \mathbf{w})}{\partial x_i \partial x_j}\epsilon_j\right) + O(\epsilon_i^2)\right)^2\right] \\
&= \frac{1}{2}\sum_{i=1}^{N}\left(\left(\frac{\partial E(\mathbf{x}; \mathbf{w})}{\partial x_i}\right)^2 + \sigma^2 \sum_{j=1}^{N}\left(\frac{\partial^2 E(\mathbf{x}; \mathbf{w})}{\partial x_i \partial x_j}\right)^2 + \hat{O}(\epsilon_i^2)\right)
\end{aligned}
\quad (4)
$$

The expected value of the second term is:

$$
\begin{aligned}
\sum_{i=1}^{N}\left(\mathbb{E}\left[\frac{\partial^2 E(\mathbf{x}+\epsilon; \mathbf{w})}{(\partial(x_i+\epsilon_i))^2}\right]\right) &= \sum_{i=1}^{N}\left(\mathbb{E}\left[\frac{\partial^2 E(\mathbf{x}; \mathbf{w})}{(\partial x_i)^2} + \sum_{i=1}^{N}\left(\frac{\partial^3 E(\mathbf{x}; \mathbf{w})}{\partial x_i \partial x_i \partial x_j}\epsilon_j\right) + O(\epsilon_i^2)\right]\right) \\
&= \sum_{i=1}^{N}\left(\frac{\partial^2 E(\mathbf{x}; \mathbf{w})}{(\partial x_i)^2}\right) + O(\epsilon_i^2)
\end{aligned}
\quad (5)
$$

Putting the terms back together, we have:

$$\mathbb{E}\left[J^S(\mathbf{x}+\epsilon; \mathbf{w})\right] = \frac{1}{2}\sum_{i=1}^{N}\left(\frac{\partial E}{\partial x_i}\right)^2 - \sum_{i=1}^{N}\left(\frac{\partial^2 E}{(\partial x_i)^2}\right) + \frac{1}{2}\sigma^2 \sum_{i=1}^{N}\sum_{j=1}^{N}\left(\frac{\partial^2 E}{\partial x_i \partial x_j}\right)^2 + \hat{O}(\epsilon^2) \quad (6)$$

where $E = E(\mathbf{x}; \mathbf{w})$. This is the full regularized Score Matching loss. While minimization of above loss may be feasible in some situations, in general it requires differentiation of the full Hessian w.r.t. $\mathbf{x}$ which scales like $O(W^2)$. However, the off-diagonal elements of the Hessian are often dominated by the diagonal. Therefore, we will use the diagonal approximation:

$$J_{reg}(\mathbf{x}; \mathbf{w}; \lambda) = J^S(\mathbf{x}; \mathbf{w}) + \lambda \sum_{i=1}^{N}\left(\frac{\partial^2 E}{(\partial x_i)^2}\right)^2 \quad (7)$$

where $\lambda$ sets regularization strength and is related to (but not exactly equal to) $\frac{1}{2}\sigma^2$ in equation (6). This regularized loss is computationally convenient: the added complexity is almost negligible since differentiation of the second derivative terms $(\partial^2 E/(\partial x_i)^2)$ w.r.t. the weights is already required for unregularized Score Matching. The regularizer is related to Tikhonov regularization [22] and *curvature-driven smoothing* [2] where the square of the curvature of the energy surface at the data points are also penalized. However, its application has been limited since (contrary to our case) in the general case it adds considerable computational cost.

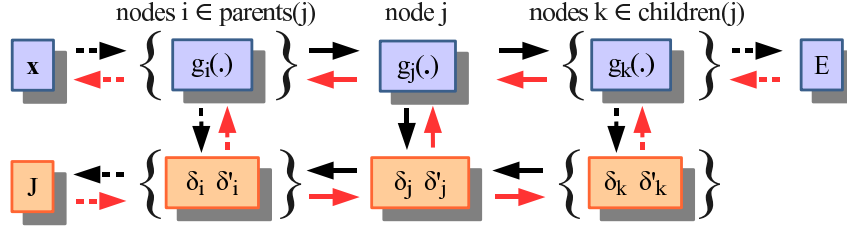

Figure 1: Illustration of local computational flow around some node $j$. Black lines: computation of quantities $\delta_j = \partial E/\partial g_j$, $\delta'_j = \partial^2 E/(\partial g_i)^2$ and the SM loss $J(\mathbf{x}; \mathbf{w})$. Red lines indicate computational flow for differentiation of the Score Matching loss: computation of e.g. $\partial J/\partial \delta_j$ and $\partial J/\partial g_j$. The influence of weights are not shown, for which the derivatives are computed in the last step.

# 3 Automatic differentiation of $J(\mathbf{x}; \mathbf{w})$

In most optimization methods for energy-based models [17], the sample loss is defined in readily obtainable quantities obtained by forward inference in the model. In such situations, the required derivatives w.r.t. the weights can be obtained in a straightforward and efficient fashion by standard application of the backpropagation algorithm.

For Score Matching, the situation is more complex since the (regularized) loss (equations 2,7) is defined in terms of $\{\partial E/\partial x_i\}$ and $\{\partial^2 E/(\partial x_i)^2\}$, each term being some function of $\mathbf{x}$ and $\mathbf{w}$. In earlier publications on Score Matching for continuous variables [10, 12, 13, 15], the authors rewrote $\{\partial E/\partial x_i\}$ and $\{\partial^2 E/(\partial x_i)^2\}$ to their explicit forms in terms of $\mathbf{x}$ and $\mathbf{w}$ by manually differentiating the energy[2]. Subsequently, derivatives of the loss w.r.t. the weights can be found. This manual differentiation was repeated for different models, and is arguably a rather inflexible approach. A procedure that could automatically (1) compute and (2) differentiate the loss would make SM estimation more accessible and flexible in practice.

A large class of models (e.g. ICA, Product-of-Experts and Fields-of-Experts), can be interpreted as a form of feed-forward neural network. Consequently, the terms $\{\partial E/\partial x_i\}$ and $\{\partial^2 E/(\partial x_i)^2\}$ can be efficiently computed using a forward and backward pass: the first pass performs forward inference (computation of $E(\mathbf{x}; \mathbf{w})$) and the second pass applies the backpropagation algorithm [3] to obtain the derivatives of the energy w.r.t. the data point ($\{\partial E/\partial x_i\}$ and $\{\partial^2 E/(\partial x_i)^2\}$). However, only the *loss* $J(\mathbf{x}; \mathbf{w})$ is obtained by these two steps. For *differentiation* of this loss, one must perform an *additional* forward and backward pass.

## 3.1 Obtaining the loss

Consider a feed-forward neural network with input vector $\mathbf{x}$ and weights $\mathbf{w}$ and an ordered set of nodes indexed $1 \ldots N$, each node $j$ with child nodes $i \in children(j)$ with $j < i$ and parent nodes $k \in parents(j)$ with $k < j$. The first $D < N$ nodes are input nodes, for which the activation value is $g_j = x_j$. For the other nodes (hidden units and output unit), the activation value is determined by a differentiable scalar function $g_j(\{g_i\}_{i \in parents(j)}, \mathbf{w})$. The network's "output" (energy) is determined as the activation of the last node: $E(\mathbf{x}; \mathbf{w}) = g_N(.)$. The values $\delta_j = \partial E/\partial g_j$ are efficiently computed by backpropagation. However, backpropagation of the full Hessian scales like $O(W^2)$, where $W$ is the number of model weights. Here, we limit backpropagation to the diagonal approximation which scales like $O(W)$ [1]. This will still result in the correct gradients $\partial^2 E/(\partial x_j)^2$ for one-layer models and the models considered in this paper. Rewriting the equations for the full Hessian is a straightforward exercise. For brevity, we write $\delta'_j = \partial^2 E/(\partial g_j)^2$. The SM loss is split in two terms: $J(\mathbf{x}; \mathbf{w}) = K + L$ with $K = \frac{1}{2}\sum_{j=1}^{D} \delta_j^2$ and $L = \sum_{j=1}^{D} -\delta'_j + \lambda(\delta'_j)^2$. The equations for inference and backpropagation are given as the first two $for$-loops in Algorithm 1.

**Input:** $\mathbf{x}, \mathbf{w}$ (data and weight vectors)

**for** $j \leftarrow D+1$ **to** $N$ **do**       // Forward propagation
 compute $g_j(.)$
 **for** $i \in parents(j)$ **do**
  compute $\frac{\partial g_j}{\partial g_i}, \frac{\partial^2 g_j}{(\partial g_i)^2}, \frac{\partial^3 g_j}{(\partial g_i)^3}$

$\delta_N \leftarrow 1, \delta'_N \leftarrow 0$
**for** $j \leftarrow N-1$ **to** $1$ **do**       // Backpropagation
 $\delta_j \leftarrow \sum_{k \in children(j)} \delta_k \frac{\partial g_k}{\partial g_j}$
 $\delta'_j \leftarrow \sum_{k \in children(j)} \delta_k \frac{\partial^2 g_k}{(\partial g_j)^2} + \delta'_k \left( \frac{\partial g_k}{\partial g_j} \right)^2$

**for** $j \leftarrow 1$ **to** $D$ **do**
 $\frac{\partial K}{\partial \delta_j} \leftarrow \delta_j; \frac{\partial L}{\partial \delta_j} \leftarrow 0; \frac{\partial L}{\partial \delta'_j} \leftarrow -1 + 2\lambda \delta'_j$

**for** $j \leftarrow D+1$ **to** $N$ **do**       // SM Forward propagation
 $\frac{\partial K}{\partial \delta_j} \leftarrow \sum_{i \in parents(j)} \frac{\partial K}{\partial \delta_i} \frac{\partial g_j}{\partial g_i}$
 $\frac{\partial L}{\partial \delta_j} \leftarrow \sum_{i \in parents(j)} \frac{\partial L}{\partial \delta'_i} \frac{\partial^2 g_j}{(\partial g_i)^2} + \frac{\partial L}{\partial \delta_i} \frac{\partial g_j}{\partial g_i}$
 $\frac{\partial L}{\partial \delta'_j} \leftarrow \sum_{i \in parents(j)} \frac{\partial L}{\partial \delta'_i} \left( \frac{\partial g_j}{\partial g_i} \right)^2$

**for** $j \leftarrow N$ **to** $D+1$ **do**       // SM Backward propagation
 $\frac{\partial K}{\partial g_j} \leftarrow \sum_{k \in children(j)} \frac{\partial K}{\partial g_k} \frac{\partial g_k}{\partial g_j} + \frac{\partial K}{\partial \delta_j} \delta_k \frac{\partial^2 g_k}{(\partial g_j)^2}$
 $\frac{\partial L}{\partial g_j} \leftarrow \sum_{k \in children(j)} \frac{\partial L}{\partial g_k} \frac{\partial g_k}{\partial g_j} + \frac{\partial L}{\partial \delta_j} \delta_k \frac{\partial^2 g_k}{(\partial g_j)^2} + 2\frac{\partial L}{\partial \delta'_j} \delta'_k \frac{\partial g_k}{\partial g_j} \frac{\partial^2 g_k}{(\partial g_j)^2} + \frac{\partial L}{\partial \delta'_j} \delta_k \frac{\partial^3 g_k}{(\partial g_j)^3}$

**for** $w \in \mathbf{w}$ **do**       // Derivatives wrt weights
 $\frac{\partial J}{\partial w} \leftarrow \sum_{j=D+1}^{N} \frac{\partial K}{\partial g_j} \frac{\partial g_j}{\partial w} + \frac{\partial L}{\partial g_j} \frac{\partial g_j}{\partial w} + \frac{\partial K}{\partial \delta_j} \frac{\partial \delta_j}{\partial w} + \frac{\partial L}{\partial \delta_j} \frac{\partial \delta_j}{\partial w} + \frac{\partial L}{\partial \delta'_j} \frac{\partial \delta'_j}{\partial w}$

**Algorithm 1: Compute** $\nabla_{\mathbf{w}} J$. See sections 3.1 and 3.2 for context.

### 3.2 Differentiating the loss

Since the computation of the loss $J(\mathbf{x}; \mathbf{w})$ is performed by a deterministic *forward-backward* mechanism, this two-step computation can be interpreted as a combination of two networks: the original network for computing $\{g_j\}$ and $E(\mathbf{x}; \mathbf{w})$, and an appended network for computing $\{\delta_j\}$, $\{\delta'_j\}$ and eventually $J(\mathbf{x}; \mathbf{w})$. See figure 1. The combined network can be differentiated by an extended version of the *double-backpropagation* procedure [5], with the main difference that the appended network not only computes $\{\delta_j\}$, but also $\{\delta'_j\}$. Automatic differentiation of the combined network consists of two phases, corresponding to reverse traversal of the appended and original network respectively: (1) obtaining $\partial K / \partial \delta_j$, $\partial L / \partial \delta_j$ and $\partial L / \partial \delta'_j$ for each node $j$ in order 1 to $N$; (2) obtaining $\partial J / \partial g_j$ for each node $j$ in order $N$ to $D+1$. These procedures are given as the last two *for*-loops in Algorithm 1. The complete algorithm scales like $O(W)$.

## 4 Experiments

Consider the following *Product-of-Experts* (PoE) model:

$$E(\mathbf{x}; W, \boldsymbol{\alpha}) = \sum_{i=1}^{M} \alpha_i g(\mathbf{w}_i^T \mathbf{x}) \tag{8}$$

where $M$ is the number of experts, $\mathbf{w}_i$ is an image filter and the $i$-th row of $W$ and $\alpha_i$ are scaling parameters. Like in [10], the filters are L2 normalized to prevent a large portion from vanishing. We use a slightly modified Student's $t$-distribution ($g(z) = log((cz)^2/2 + 1)$) for latent space, so this is also a *Product of Student's $t$-distribution* model [24]. The parameter $c$ is a non-learnable horizontal scaling parameter, set to $e^{1.5}$. The vertical scaling parameters $\alpha_i$ are restricted to positive, by setting $\alpha_i = \exp \beta_i$ where $\beta_i$ is the actual weight.

### 4.1 MNIST

The first task is to estimate a density model of the MNIST handwritten digits [16]. Since a large number of models need to be learned, a $2\times$ downsampled version of MNIST was used. The MNIST dataset is highly non-smooth: for each pixel, the extreme values (0 and 1) are highly frequent leading to sharp discontinuities in the data density at these points. It is well known that for models with square weight matrix $W$, normalized $g(.)$ (meaning $\int_{-\infty}^{\infty} \exp(-g(x))dx = 1$) and $\alpha_i = 1$, the normalizing constant can be computed [10]: $Z(\mathbf{w}) = |\det W|$. For this special case, models can be compared by computing the log-likelihood for the training- and test set. Unregularized, and regularized models for different choices of $\lambda$ were estimated and log-likelihood values were computed. Subsequently, these models were compared on a classification task. For each MNIST digit class, a small sample of 100 data points was converted to internal features by different models. These features, combined with the original class label, were subsequently used to train a logistic regression classifier for each model. For the PoE model, the "activations" $g(\mathbf{w}_i^T \mathbf{x})$ were used as features. Classification error on the test set was compared against reported results for optimal RBM and SESM models [20].

**Results.** As expected, unregularized estimation did not result in an accurate model. Figure 2 shows how the log-likelihood of the train- and test set is optimal at $\lambda^* \approx 0.01$, and decreases for smaller $\lambda$. Coincidentally, the classification performance is optimal for the same choice of $\lambda$.

### 4.2 Denoising

Consider grayscale natural image data from the Berkeley dataset [18]. The data quantized and therefore non-smooth, so regularization is potentially beneficial. In order to estimate the correct regularization magnitude, we again esimated a PoE model as in equation (8) with square $W$, such that $Z(\mathbf{w}) = |\det W|$ and computed the log-likelihood of 10.000 random patches under different regularization levels. We found that $\lambda^* \approx 10^{-5}$ for maximum likelihood (see figure 2d). This value is lower than for MNIST data since natural image data is "less unsmooth". Subsequently, a convolutional PoE model known as Fields-of-Experts [21] (FoE) was estimated using regularized SM:

$$E(\mathbf{x}; W, \boldsymbol{\alpha}) = \sum_p \sum_{i=1}^M \alpha_i g(\mathbf{w}_i^T \mathbf{x}_{(p)}) \tag{9}$$

where $p$ runs over image positions, and $x_{(p)}$ is a square image patch at $p$. The first model has the same architecture as the CD-1 trained model in [21]: $5 \times 5$ receptive fields, 24 experts ($M = 24$), and $\alpha_i$ and $g(.)$ as in our PoE model. Note that qualitative results of a similar model estimated with SM have been reported earlier [15]. We found that for best performance, the model is learned on images "whitened" with a $5 \times 5$ Laplacian kernel. This is approximately equivalent to ZCA whitening used in [15].

Models are evaluated by means of Bayesian denoising using *maximum a posteriori* (MAP) estimation. As in a general Bayesian image restoration framework, the goal is to estimate the original input $\mathbf{x}$ given a noisy image $\mathbf{y}$ using the Bayesian proportionality $p(\mathbf{x}|\mathbf{y}) \propto p(\mathbf{y}|\mathbf{x})p(\mathbf{x})$. The assumption is white Gaussian noise such that the likelihood is $p(\mathbf{y}|\mathbf{x}) \sim \mathcal{N}(\mathbf{0}, \sigma^2 I)$. The model $E(\mathbf{x}; \mathbf{w}) = -\log p(\mathbf{x}; \mathbf{w}) - Z(\mathbf{w})$ is our prior. The gradient of the log-posterior is:

$$\nabla_{\mathbf{x}} \log p(\mathbf{x}|\mathbf{y}) = -\nabla_{\mathbf{x}} E(\mathbf{x}; \mathbf{w}) + \frac{1}{2\sigma^2} \nabla_{\mathbf{x}} \sum_{i=1}^N (y_i - x_i)^2 \tag{10}$$

Denoising is performed by initializing $\mathbf{x}$ to a noise image, and 300 subsequent steps of steepest descent according to $\mathbf{x}' \leftarrow \mathbf{x} + \alpha \nabla_{\mathbf{x}} \log p(\mathbf{x}|\mathbf{y})$, with $\alpha$ annealed from $2 \cdot 10^{-2}$ to $5 \cdot 10^{-4}$. For comparison, we ran the same denoising procedure with models estimated by CD-1 and Basis Rotation, from [21] and [23] respectively. Note that the CD-1 model is trained using PCA whitening. The CD-1 model has been extensively applied to denoising before [21] and shown to compare favourably to specialized denoising methods.

**Results.** Training of the convolutional model took about 1 hour on a 2Ghz machine. Regularization turns out to be important for optimal denoising (see figure 2[e-g]). See table 1 for denoising performance of the optimal model for specific standard images. Our model performed significantly better

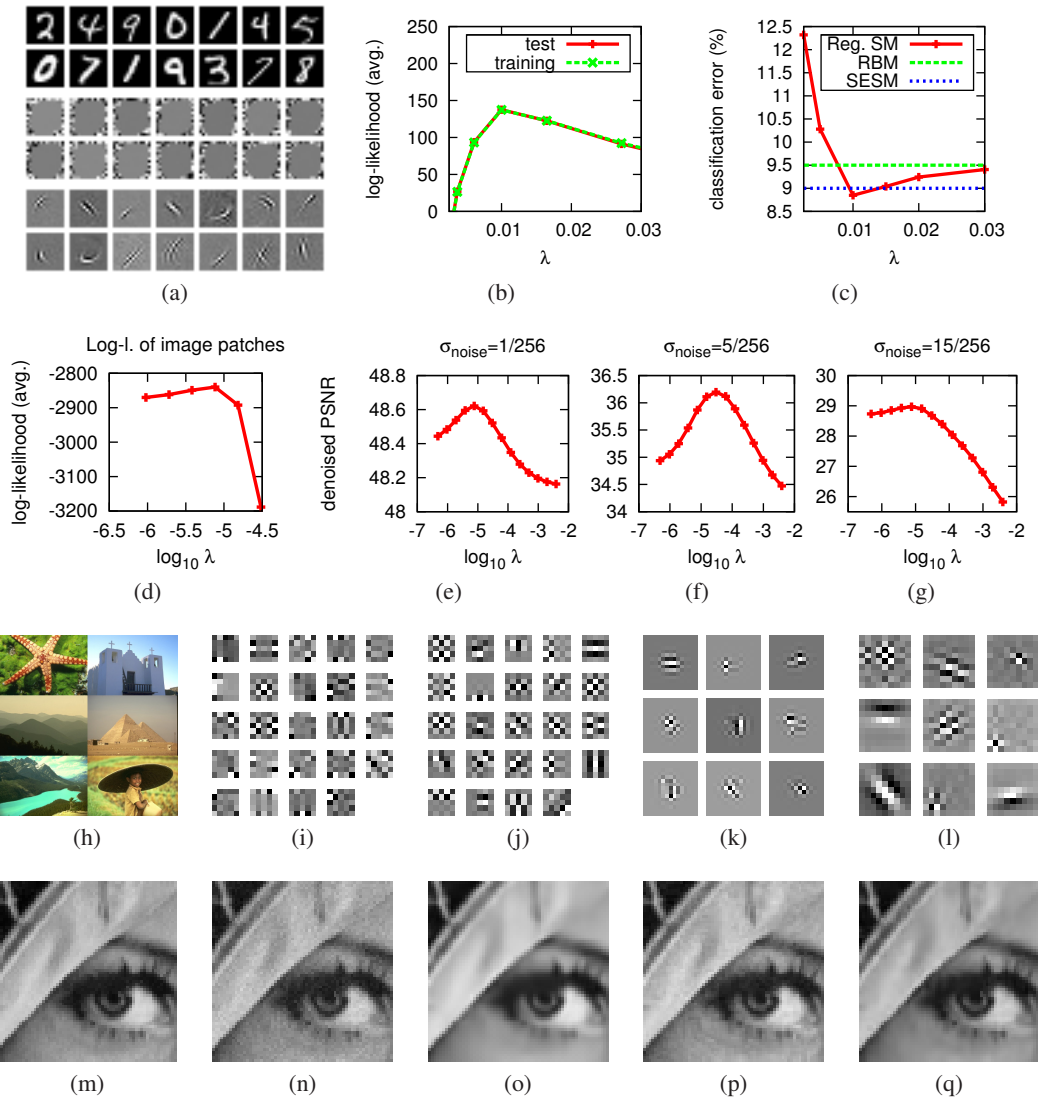

Figure 2: **(a)** Top: selection of downsampled MNIST datapoints. Middle and bottom: random sample of filters from unregularized and regularized ($\lambda = 0.01$) models, respectively. **(b)** Average log-likelihood of MNIST digits in training- and test sets for choices of $\lambda$. Note that $\lambda^* \approx 0.01$, both for maximum likelihood and optimal classification. **(c)** Test set error of a logistic regression classifier learned on top of features, with only 100 samples per class, for different choices of $\lambda$. Optimal error rates of SESM and RBM (figure 1a in [20]) are shown for comparison. **(d)** Log-likelihood of 10.000 random natural image patches for complete model, for different choices of $\lambda$. **(e-g)** PSNR of 500 denoised images, for different levels of noise and choices of $\lambda$. Note that $\lambda^* \approx 10^{-5}$, both for maximum likelihood and best denoising performance. **(h)** Some natural images from the Berkeley dataset. **(i)** Filters of model with $5 \times 5 \times 24$ weights learned with CD-1 [21], **(j)** filters of our model with $5 \times 5 \times 24$ weights, **(k)** random selection of filters from the Basis Rotation [23] model with $15 \times 15 \times 25$ weights, **(l)** random selection of filters from our model with $8 \times 8 \times 64$ weights. **(m)** Detail of original Lena image. **(n)** Detail with noise added ($\sigma_{noise} = 5/256$). **(o)** Denoised with model learned with CD-1 [21], **(p)** Basis Rotation [23], **(q)** and Score Matching with (near) optimal regularization.

than the Basis Rotation model and slightly better than the CD-1 model. As reported earlier in [15], we can verify that the filters are completely intuitive (Gabor filters with different phase, orientation and scale) unlike the filters of CD-1 and Basis Rotation models (see figure 2[i-l]).

Table 1: Peak signal-to-noise ratio (PSNR) of denoised images with $\sigma_{noise} = 5/256$. Shown errors are aggregated over different noisy images.

| Image | CD-1 | Basis Rotation | Our model |
|---|---|---|---|
| Weights | $(5 \times 5) \times 24$ | $(15 \times 15) \times 25$ | $(5 \times 5) \times 24$ |
| Barbara | **37.30**±0.01 | 37.08±0.02 | **37.31**±0.01 |
| Peppers | **37.63**±0.01 | 37.09±0.02 | 37.41±0.03 |
| House | 37.85±0.02 | 37.73±0.03 | **38.03**±0.04 |
| Lena | 38.16±0.02 | 37.97±0.01 | **38.19**±0.01 |
| Boat | 36.33±0.01 | 36.21±0.01 | **36.53**±0.01 |

### 4.3 Super-resolution

In addition, models are compared with respect to their performance on a simple version of *super-resolution* as follows. An original image $\mathbf{x}_{orig}$ is sampled down to image $\mathbf{x}_{small}$ by averaging blocks of $2 \times 2$ pixels into a single pixel. A first approximation $\mathbf{x}$ is computed by linearly scaling up $\mathbf{x}_{small}$ and subsequent application of a low-pass filter to remove false high frequency information. The image is than fine-tuned by 200 repetitions of two subsequent steps: (1) refining the image slightly using $\mathbf{x}' \leftarrow \mathbf{x} + \alpha \nabla_{\mathbf{x}} E(\mathbf{x}; \mathbf{w})$ with $\alpha$ annealed from $2 \cdot 10^{-2}$ to $5 \cdot 10^{-4}$ ; (2) updating each $k \times k$ block of pixels such that their average corresponds to the down-sampled value. Note: the simple block-downsampling results in serious aliasing artifacts in the Barbara image, so the Castle image is used instead.

**Results.** PSNR values for standard images are shown in table 2. The considered models made give slight improvements in terms of PSNR over the initial solution with low pass filter. Still, our model did slightly better than the CD-1 and Basis Rotation models.

Table 2: Peak signal-to-noise ratio (PSNR) of super-resolved images for different models.

| Image | Low pass filter | CD-1 | Basis Rotation | Our model |
|---|---|---|---|---|
| Weights | - | $(5 \times 5) \times 24$ | $(15 \times 15) \times 25$ | $(5 \times 5) \times 24$ |
| Peppers | 27.54 | 29.11 | 27.69 | **29.76** |
| House | 33.15 | **33.53** | 33.41 | 33.48 |
| Lena | 32.39 | 33.31 | 33.07 | **33.46** |
| Boat | 29.20 | 30.81 | 30.77 | **30.82** |
| Castle | 24.19 | 24.15 | 24.26 | **24.31** |

## 5 Conclusion

We have shown how the addition of a principled regularization term to the expression of the Score Matching loss lifts continuity assumptions on the data density, such that the estimation method becomes more generally applicable. The effectiveness of the regularizer was verified with the discontinuous MNIST and Berkeley datasets, with respect to likelihood of test data in the model. For both datasets, the optimal regularization parameter is approximately equal for both likelihood and subsequent classification and denoising tasks. In addition, we showed how computation and differentiation of the Score Matching loss can be automated using an efficient algorithm.

## Footnotes

[1] The conditions are: the true (underlying) pdf $p_{\mathbf{x}}(\mathbf{x})$ is differentiable, the expectations $\mathbb{E}\left[\|\partial \log p_{\mathbf{x}}(\mathbf{x})/\partial \mathbf{x}\|^2\right]$ and $\mathbb{E}\left[\|\partial E(\mathbf{x}; \mathbf{w})/\partial \mathbf{x}\|^2\right]$ w.r.t. $\mathbf{x}$ are finite for any $\mathbf{w}$, and $p_{\mathbf{x}}(\mathbf{x}) \partial E(\mathbf{x}; \mathbf{w})/\partial \mathbf{x}$ goes to zero for any $\mathbf{w}$ when $\|\mathbf{x}\| \to \infty$.

[2]Most previous publications do not express unnormalized neg. log-density as "energy"

# References

[1] S. Becker and Y. LeCun. Improving the convergence of back-propagation learning with second-order methods. In D. Touretzky, G. Hinton, and T. Sejnowski, editors, *Proc. of the 1988 Connectionist Models Summer School*, pages 29–37, San Mateo, 1989. Morgan Kaufman.

[2] C. M. Bishop. *Neural networks for pattern recognition*. Oxford University Press, Oxford, UK, 1996.

[3] A. E. Bryson and Y. C. Ho. *Applied optimal control; optimization, estimation, and control*. Blaisdell Pub. Co. Waltham, Massachusetts, 1969.

[4] M. A. Carreira-Perpinan and G. E. Hinton. On contrastive divergence learning. In *Artificial Intelligence and Statistics*, 2005.

[5] H. Drucker and Y. LeCun. Improving generalization performance using double backpropagation. *IEEE Transactions on Neural Networks*, 3(6):991–997, 1992.

[6] M. Gutmann and A. Hyvärinen. Noise-contrastive estimation: A new estimation principle for unnormalized statistical models. In *Proc. Int. Conf. on Artificial Intelligence and Statistics (AISTATS2010)*, 2010.

[7] G. E. Hinton. Training products of experts by minimizing contrastive divergence. *Neural Computation*, 14:2002, 2000.

[8] G. E. Hinton, S. Osindero, and Y. W. Teh. A fast learning algorithm for deep belief nets. *Neural Computation*, 18(7):1527–1554, 2006.

[9] G. E. Hinton, S. Osindero, M. Welling, and Y. W. Teh. Unsupervised discovery of non-linear structure using contrastive backpropagation. *Cognitive Science*, 30(4):725–731, 2006.

[10] A. Hyvärinen. Estimation of non-normalized statistical models by score matching. *Journal of Machine Learning Research*, 6:695–709, 2005.

[11] A. Hyvärinen. Some extensions of score matching. *Computational Statistics & Data Analysis*, 51(5):2499–2512, 2007.

[12] A. Hyvärinen. Optimal approximation of signal priors. *Neural Computation*, 20:3087–3110, 2008.

[13] U. Köster and A. Hyvärinen. A two-layer ica-like model estimated by score matching. In J. M. de Sá, L. A. Alexandre, W. Duch, and D. P. Mandic, editors, *ICANN (2)*, volume 4669 of *Lecture Notes in Computer Science*, pages 798–807. Springer, 2007.

[14] U. Koster, J. T. Lindgren, and A. Hyvärinen. Estimating markov random field potentials for natural images. *Proc. Int. Conf. on Independent Component Analysis and Blind Source Separation (ICA2009)*, 2009.

[15] U. Köster, J. T. Lindgren, and A. Hyvärinen. Estimating markov random field potentials for natural images. In T. Adali, C. Jutten, J. M. T. Romano, and A. K. Barros, editors, *ICA*, volume 5441 of *Lecture Notes in Computer Science*, pages 515–522. Springer, 2009.

[16] Y. LeCun, L. Bottou, Y. Bengio, and P. Haffner. Gradient-based learning applied to document recognition. In *Proceedings of the IEEE*, pages 2278–2324, 1998.

[17] Y. LeCun, S. Chopra, R. Hadsell, M. Ranzato, and F. Huang. A tutorial on energy-based learning. In G. Bakir, T. Hofman, B. Schölkopf, A. Smola, and B. Taskar, editors, *Predicting Structured Data*. MIT Press, 2006.

[18] D. Martin, C. Fowlkes, D. Tal, and J. Malik. A database of human segmented natural images and its application to evaluating segmentation algorithms and measuring ecological statistics. In *Proc. 8th Int'l Conf. Computer Vision*, volume 2, pages 416–423, July 2001.

[19] S. Osindero and G. E. Hinton. Modeling image patches with a directed hierarchy of markov random fields. In J. Platt, D. Koller, Y. Singer, and S. Roweis, editors, *Advances in Neural Information Processing Systems 20*, pages 1121–1128. MIT Press, Cambridge, MA, 2008.

[20] M. Ranzato, Y. Boureau, and Y. LeCun. Sparse feature learning for deep belief networks. In *Advances in Neural Information Processing Systems (NIPS 2007)*, 2007.

[21] S. Roth and M. J. Black. Fields of experts. *International Journal of Computer Vision*, 82(2):205–229, 2009.

[22] A. N. Tikhonov. On the stability of inverse problems. *Dokl. Akad. Nauk SSSR*, (39):176–179, 1943.

[23] Y. Weiss and W. T. Freeman. What makes a good model of natural images. In *CVPR 2007: Proceedings of the 2007 IEEE Computer Society Conference on Computer Vision and Pattern Recognition, IEEE Computer Society*, pages 1–8, 2007.

[24] M. Welling, G. E. Hinton, and S. Osindero. Learning sparse topographic representations with products of student-t distributions. In S. T. S. Becker and K. Obermayer, editors, *Advances in Neural Information Processing Systems 15*, pages 1359–1366. MIT Press, Cambridge, MA, 2003.

